# Sparse deep belief net model for visual area V2

**Honglak Lee**      **Chaitanya Ekanadham**      **Andrew Y. Ng**
Computer Science Department
Stanford University
Stanford, CA 94305
{hllee,chaitu,ang}@cs.stanford.edu

## Abstract

Motivated in part by the hierarchical organization of the cortex, a number of algorithms have recently been proposed that try to learn hierarchical, or "deep," structure from unlabeled data. While several authors have formally or informally compared their algorithms to computations performed in visual area V1 (and the cochlea), little attempt has been made thus far to evaluate these algorithms in terms of their fidelity for mimicking computations at deeper levels in the cortical hierarchy. This paper presents an unsupervised learning model that faithfully mimics certain properties of visual area V2. Specifically, we develop a sparse variant of the deep belief networks of Hinton et al. (2006). We learn two layers of nodes in the network, and demonstrate that the first layer, similar to prior work on sparse coding and ICA, results in localized, oriented, edge filters, similar to the Gabor functions known to model V1 cell receptive fields. Further, the second layer in our model encodes correlations of the first layer responses in the data. Specifically, it picks up both colinear ("contour") features as well as corners and junctions. More interestingly, in a quantitative comparison, the encoding of these more complex "corner" features matches well with the results from the Ito & Komatsu's study of biological V2 responses. This suggests that our sparse variant of deep belief networks holds promise for modeling more higher-order features.

## 1 Introduction

The last few years have seen significant interest in "deep" learning algorithms that learn layered, hierarchical representations of high-dimensional data. [1, 2, 3, 4]. Much of this work appears to have been motivated by the hierarchical organization of the cortex, and indeed authors frequently compare their algorithms' output to the oriented simple cell receptive fields found in visual area V1. (E.g., [5, 6, 2]) Indeed, some of these models are often viewed as first attempts to elucidate what learning algorithm (if any) the cortex may be using to model natural image statistics.

However, to our knowledge no serious attempt has been made to directly relate, such as through quantitative comparisons, the computations of these deep learning algorithms to areas deeper in the cortical hierarchy, such as to visual areas V2, V4, etc. In this paper, we develop a sparse variant of Hinton's deep belief network algorithm, and measure the degree to which it faithfully mimics biological measurements of V2. Specifically, we take Ito & Komatsu [7]'s characterization of V2 in terms of its responses to a large class of angled bar stimuli, and quantitatively measure the degree to which the deep belief network algorithm generates similar responses.

Deep architectures attempt to learn hierarchical structure, and hold the promise of being able to first learn simple concepts, and then successfully build up more complex concepts by composing together the simpler ones. For example, Hinton et al. [1] proposed an algorithm based on learning individual layers of a hierarchical probabilistic graphical model from the bottom up. Bengio et al. [3] proposed a similarly greedy algorithm, one based on autoencoders. Ranzato et al. [2] developed an energy-based hierarchical algorithm, based on a sequence of sparsified autoencoders/decoders.

In related work, several studies have compared models such as these, as well as non-hierarchical/non-deep learning algorithms, to the response properties of neurons in area V1. A study by van Hateren and van der Schaaf [8] showed that the filters learned by independent components analysis (ICA) [9] on natural image data match very well with the classical receptive fields of V1 simple cells. (Filters learned by sparse coding [10, 11] also similarly give responses similar to V1 simple cells.) Our work takes inspiration from the work of van Hateren and van der Schaaf, and represents a study that is done in a similar spirit, only extending the comparisons to a deeper area in the cortical hierarchy, namely visual area V2.

## 2 Biological comparison

### 2.1 Features in early visual cortex: area V1

The selectivity of neurons for oriented bar stimuli in cortical area V1 has been well documented [12, 13]. The receptive field of simple cells in V1 are localized, oriented, bandpass filters that resemble gabor filters. Several authors have proposed models that have been either formally or informally shown to replicate the gabor-like properties of V1 simple cells. Many of these algorithms, such as [10, 9, 8, 6], compute a (approximately or exactly) sparse representation of the natural stimuli data. These results are consistent with the "efficient coding hypothesis" which posits that the goal of early visual processing is to encode visual information as efficiently as possible [14]. Some hierarchical extensions of these models [15, 6, 16] are able to learn features that are more complex than simple oriented bars. For example, hierarchical sparse models of natural images have accounted for complex cell receptive fields [17], topography [18, 6], colinearity and contour coding [19]. Other models, such as [20], have also been shown to give V1 complex cell-like properties.

### 2.2 Features in visual cortex area V2

It remains unknown to what extent the previously described algorithms can learn higher order features that are known to be encoded further down the ventral visual pathway. In addition, the response properties of neurons in cortical areas receiving projections from area V1 (e.g., area V2) are not nearly as well documented. It is uncertain what type of stimuli cause V2 neurons to respond optimally [21]. One V2 study by [22] reported that the receptive fields in this area were similar to those in the neighboring areas V1 and V4. The authors interpreted their findings as suggestive that area V2 may serve as a place where different channels of visual information are integrated. However, quantitative accounts of responses in area V2 are few in number. In the literature, we identified two sets of quantitative data that give us a good starting point for making measurements to determine whether our algorithms may be computing similar functions as area V2.

In one of these studies, Ito and Komatsu [7] investigated how V2 neurons responded to angular stimuli. They summarized each neuron's response with a two-dimensional visualization of the stimuli set called an angle profile. By making several axial measurements within the profile, the authors were able to compute various statistics about each neuron's selectivity for angle width, angle orientation, and for each separate line component of the angle (see Figure 1). Approximately 80% of the neurons responded to specific angle stimuli. They found neurons that were selective for only one line component of its peak angle as well as neurons selective for both line components. These neurons yielded angle profiles resembling those of Cell 2 and Cell 5 in Figure 1, respectively. In addition, several neurons exhibited a high amount of selectivity for its peak angle producing angle profiles like that of Cell 1 in Figure 1. No neurons were found that had more elongation in a diagonal axis than in the horizontal or vertical axes, indicating that neurons in V2 were not selective for angle width or orientation. Therefore, an important conclusion made from [7] was that a V2 neuron's response to an angle stimulus is highly dependent on its responses to each individual line component of the angle. While the dependence was often observed to be simply additive, as was the case with neurons yielding profiles like those of Cells 1 and 2 in Figure 1(right), this was not always the case. 29 neurons had very small peak response areas and yielded profiles like that of Cell 1 in Figure 1(right), thus indicating a highly specific tuning to an angle stimulus. While the former responses suggest a simple linear computation of V1 neural responses, the latter responses suggest a nonlinear computation [21]. The analysis methods adopted in [7] are very useful in characterizing the response properties, and we use these methods to evaluate our own model.

Another study by Hegde and Van Essen [23] studied the responses of a population of V2 neurons to complex contour and grating stimuli. They found several V2 neurons responding maximally for angles, and the distribution of peak angles for these neurons is consistent with that found by [7]. In addition, several V2 neurons responded maximally for shapes such as intersections, tri-stars, five-point stars, circles, and arcs of varying length.

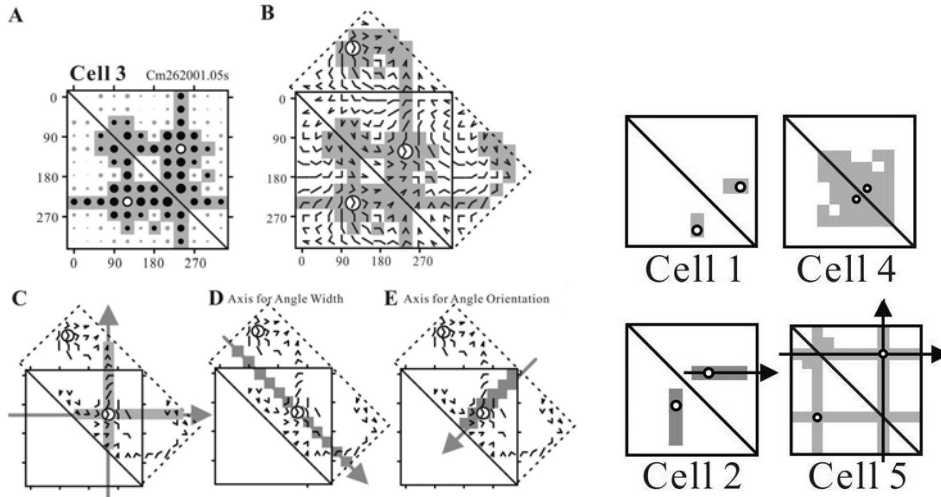

Figure 1: (Images from [7]; courtesy of Ito and Komatsu) Left: Visualization of angle profiles. The upper-right and lower-left triangles contain the same stimuli. (A,B) Darkened squares correspond to stimuli that elicited a large response. The peak responses are circled. (C) The arrangement of the figure is so that one line component remains constant as one moves along any vertical or horizontal axis. (D) The angles width remains constant as one moves along a the diagonal indicated (E) The angle orientation remains constant as one moves along the diagonal indicated. After identifying the optimal stimuli for a neuron in the profile, the number of stimuli along these various axes (as in C,D,E) eliciting responses larger than 80% of the peak response measure the neuron's tolerance to perturbations to the line components, peak angle width, and orientation, respectively. Right: Examples of 4 typical angle profiles. As before, stimuli eliciting large responses are highlighted. Cell 1 has a selective response to a stimulus, so there is no elongation along any axis. Cell 2 has one axis of elongation, indicating selectivity for one orientation. Cell 5 has two axes of elongation, and responds strongly so long as either of two edge orientations is present. Cell 4 has no clear axis of elongation.

## 3 Algorithm

Hinton et al. [1] proposed an algorithm for learning deep belief networks, by treating each layer as a restricted Boltzmann machine (RBM) and greedily training the network one layer at a time from the bottom up [24, 1]. In general, however, RBMs tend to learn distributed, non-sparse representations. Based on results from other methods (e.g., sparse coding [10, 11], ICA [9], heavy-tailed models [6], and energy based models [2]), sparseness seems to play a key role in learning gabor-like filters. Therefore, we modify Hinton et al.'s learning algorithm to enable deep belief nets to learn sparse representations.

### 3.1 Sparse restricted Boltzmann machines

We begin by describing the restricted Boltzmann machine (RBM), and present a modified version of it. An RBM has a set of hidden units $\mathbf{h}$, a set of visible units $\mathbf{v}$, and symmetric connections weights between these two layers represented by a weight matrix $W$. Suppose that we want to model $k$ dimensional real-valued data using an undirected graphical model with $n$ binary hidden units. The negative log probability of any state in the RBM is given by the following energy function:[1]

$$-\log P(\mathbf{v}, \mathbf{h}) = E(\mathbf{v}, \mathbf{h}) = \frac{1}{2\sigma^2}\sum_i v_i^2 - \frac{1}{\sigma^2}\left(\sum_i c_i v_i + \sum_j b_j h_j + \sum_{i,j} v_i w_{ij} h_j\right). \quad (1)$$

Here, $\sigma$ is a parameter, $h_j$ are hidden unit variables, $v_i$ are visible unit variables. Informally, the maximum likelihood parameter estimation problem corresponds to learning $w_{ij}, c_i$ and $b_j$ so as to minimize the energy of states drawn from the data distribution, and raise the energy of states that are improbable given the data.

Under this model, we can easily compute the conditional probability distributions. Holding either $\mathbf{h}$ or $\mathbf{v}$ fixed, we can sample from the other as follows:

$$P(v_i|\mathbf{h}) = \mathcal{N}\left(c_i + \sum_j w_{ij} h_j, \sigma^2\right), \quad (2)$$

$$P(h_j|\mathbf{v}) = logistic\left(\frac{1}{\sigma^2}\left(b_j + \sum_i w_{ij} v_i\right)\right). \quad (3)$$

Here, $\mathcal{N}(\cdot)$ is the gaussian density, and $logistic(\cdot)$ is the logistic function.

For training the parameters of the model, the objective is to maximize the log-likelihood of the data. We also want hidden unit activations to be sparse; thus, we add a regularization term that penalizes a deviation of the expected activation of the hidden units from a (low) fixed level $p$.[2] Thus, given a training set $\{\mathbf{v}^{(1)}, \dots, \mathbf{v}^{(m)}\}$ comprising $m$ examples, we pose the following optimization problem:

$$\text{minimize}_{\{w_{ij}, c_i, b_j\}} \quad -\sum_{l=1}^{m} \log \sum_{\mathbf{h}} P(\mathbf{v}^{(l)}, \mathbf{h}^{(l)}) + \lambda \sum_{j=1}^{n} |p - \tfrac{1}{m} \sum_{l=1}^{m} \mathbb{E}[h_j^{(l)} | \mathbf{v}^{(l)}]|^2, \quad (4)$$

where $\mathbb{E}[\cdot]$ is the conditional expectation given the data, $\lambda$ is a regularization constant, and $p$ is a constant controlling the sparseness of the hidden units $h_j$. Thus, our objective is the sum of a log-likelihood term and a regularization term. In principle, we can apply gradient descent to this problem; however, computing the gradient of the log-likelihood term is expensive. Fortunately, the contrastive divergence learning algorithm gives an efficient approximation to the gradient of the log-likelihood [25]. Building upon this, on each iteration we can apply the contrastive divergence update rule, followed by one step of gradient descent using the gradient of the regularization term.[3] The details of our procedure are summarized in Algorithm 1.

---

**Algorithm 1** Sparse RBM learning algorithm

1. Update the parameters using contrastive divergence learning rule. More specifically,
$$w_{ij} := w_{ij} + \alpha(\langle v_i h_j \rangle_{\text{data}} - \langle v_i h_j \rangle_{\text{recon}})$$
$$c_i := c_i + \alpha(\langle v_i \rangle_{\text{data}} - \langle v_i \rangle_{\text{recon}})$$
$$b_j := b_j + \alpha(\langle b_j \rangle_{\text{data}} - \langle b_j \rangle_{\text{recon}}),$$
where $\alpha$ is a learning rate, and $\langle \cdot \rangle_{\text{recon}}$ is an expectation over the reconstruction data, estimated using one iteration of Gibbs sampling (as in Equations 2,3).
2. Update the parameters using the gradient of the regularization term.
3. Repeat Steps 1 and 2 until convergence.

---

### 3.2   Learning deep networks using sparse RBM

Once a layer of the network is trained, the parameters $w_{ij}, b_j, c_i$'s are frozen and the hidden unit values given the data are inferred. These inferred values serve as the "data" used to train the next higher layer in the network. Hinton et al. [1] showed that by repeatedly applying such a procedure, one can learn a multilayered deep belief network. In some cases, this iterative "greedy" algorithm can further be shown to be optimizing a variational bound on the data likelihood, if each layer has at least as many units as the layer below (although in practice this is not necessary to arrive at a desirable solution; see [1] for a detailed discussion). In our experiments using natural images, we learn a network with two hidden layers, with each layer learned using the sparse RBM algorithm described in Section 3.1.

## 4   Visualization

### 4.1   Learning "strokes" from handwritten digits

We applied the sparse RBM algorithm to the MNIST handwritten digit dataset.[4] We learned a sparse RBM with 69 visible units and 200 hidden units. The learned bases are shown in Figure 2. (Each basis corresponds to one column of the weight matrix $W$ left-multiplied by the unwhitening matrix.) Many bases found by the algorithm roughly represent different "strokes" of which handwritten digits are comprised. This is consistent

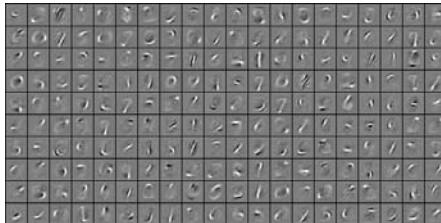

Figure 2: Bases learned from MNIST data

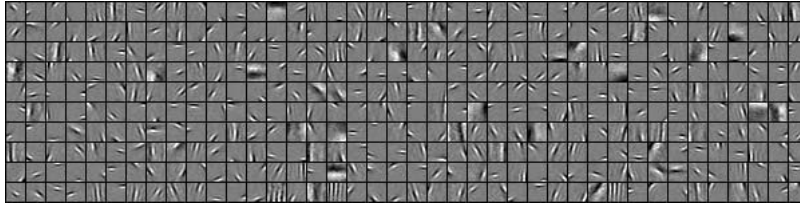

Figure 3: 400 first layer bases learned from the van Hateren natural image dataset, using our algorithm.

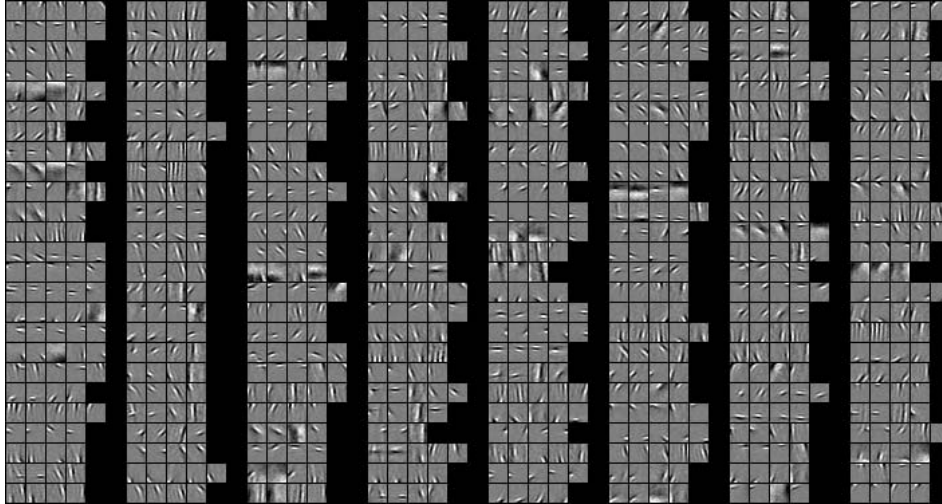

Figure 4: Visualization of 200 second layer bases (model V2 receptive fields), learned from natural images. Each small group of 3-5 (arranged in a row) images shows one model V2 unit; the leftmost patch in the group is a visualization of the model V2 basis, and is obtained by taking a weighted linear combination of the first layer "V1" bases to which it is connected. The next few patches in the group show the first layer bases that have the strongest weight connection to the model V2 basis.

with results obtained by applying different algorithms to learn sparse representations of this data set (e.g., [2, 5]).

## 4.2 Learning from natural images

We also applied the algorithm to a training set a set of 14-by-14 natural image patches, taken from a dataset compiled by van Hateren.[5] We learned a sparse RBM model with 196 visible units and 400 hidden units. The learned bases are shown in Figure 3; they are oriented, gabor-like bases and resemble the receptive fields of V1 simple cells.[6]

## 4.3 Learning a two-layer model of natural images using sparse RBMs

We further learned a two-layer network by stacking one sparse RBM on top of another (see Section 3.2 for details.)[7] After learning, the second layer weights were quite sparse—most of the weights were very small, and only a few were either highly positive or highly negative. Positive

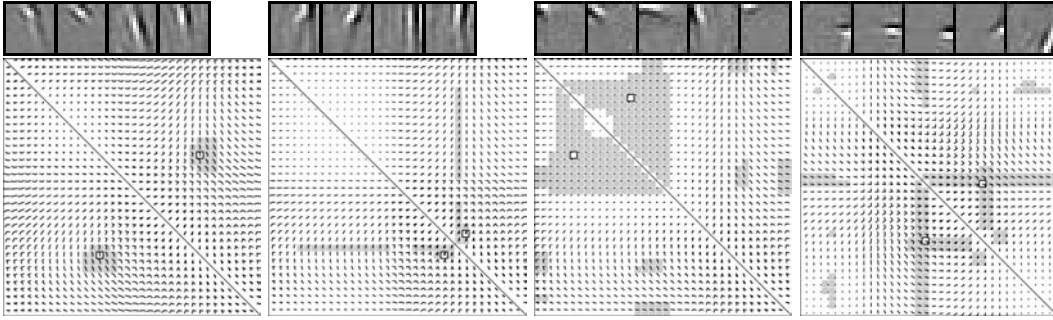

Figure 5: Top: Visualization of four learned model V2 neurons. (Visualization in each row of four or five patches follows format in Figure 4.) Bottom: Angle stimulus response profile for model V2 neurons in the top row. The 36*36 grid of stimuli follows [7], in which the orientation of two lines are varied to form different angles. As in Figure 1, darkened patches represent stimuli to which the model V2 neuron responds strongly; also, a small black square indicates the overall peak response.

weights represent excitatory connections between model V1 and model V2 units, whereas negative elements represent inhibitory connections. By visualizing the second layer bases as shown in Figure 4, we observed bases that encoded co-linear first layer bases as well as edge junctions. This shows that by extending the sparse RBM to two layers and using greedy learning, the model is able to learn bases that encode contours, angles, and junctions of edges.

## 5 Evaluation experiments

We now more quantitatively compare the algorithm's learned responses to biological measurements.[8]

### 5.1 Method: Ito-Komatsu paper protocol

We now describe the procedure we used to compare our model with the experimental data in [7]. We generated a stimulus set consisting of the same set of angles (pairs of edges) as [7]. To identify the "center" of each model neuron's receptive field, we translate all stimuli densely over the 14x14 input image patch, and identify the position at which the maximum response is elicited. All measures are then taken with all angle stimuli centered at this position.[9]

Using these stimuli, we compute the hidden unit probabilities from our model V1 and V2 neurons. In other words, for each stimulus we compute the first hidden layer activation probabilities, then feed this probability as data to the second hidden layer and compute the activation probabilities again in the same manner. Following a protocol similar to [7], we also eliminate from consideration the model neurons that do not respond strongly to corners and edges.[10] Some representative results are shown in Figure 5. (The four angle profiles shown are fairly typical of those obtained in our experiments.) We see that all the V2 bases in Figure 5 have maximal response when its strongest V1-basis components are aligned with the stimulus. Thus, some of these bases do indeed seem to encode edge junctions or crossings.

We also compute similar summary statistics as [7] (described in Figure 1(C,D,E)), that more quantitatively measure the distribution of V2 or model V2 responses to the different angle stimuli. Figure 6 plots the responses of our model, together with V2 data taken from [7]. Along many dimensions, the results from our model match that from the Macaque V2 fairly well.

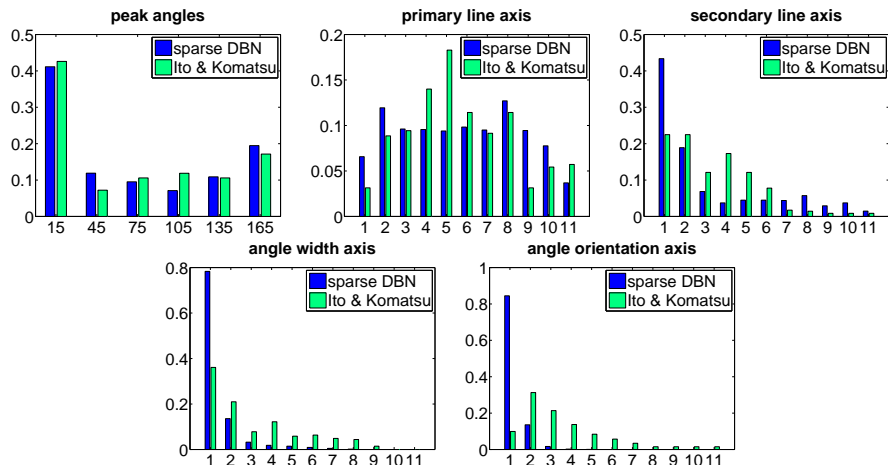

Figure 6: Images show distributions over stimulus response statistics (averaged over 10 trials) from our algorithm (blue) and in data taken from [7] (green). The five figures show respectively (i) the distribution over peak angle response (ranging from 0 to 180 degrees; each bin represents a range of 30 degrees), (ii) distribution over tolerance to primary line component (Figure 1C, in dominant vertical or horizontal direction), (iii) distribution over tolerance to secondary line component (Figure 1C, in non-dominant direction), (iv) tolerance to angle width (Figure 1D), (v) tolerance to angle orientation (Figure 1E). See Figure 1 caption, and [7], for details.

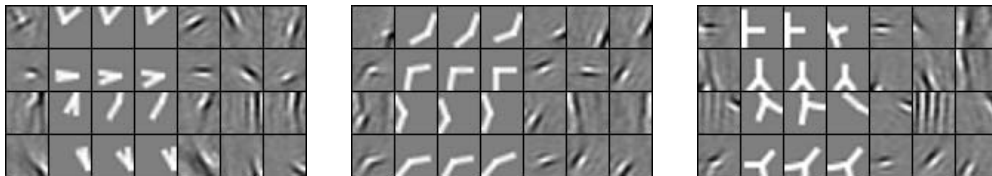

Figure 7: Visualization of a number of model V2 neurons that maximally respond to various complex stimuli. Each row of seven images represents one V2 basis. In each row, the leftmost image shows a linear combination of the top three weighted V1 components that comprise the V2 basis; the next three images show the top three optimal stiimuli; and the last three images show the top three weighted V1 bases. The V2 bases shown in the figures maximally respond to acute angles (left), obtuse angles (middle), and tri-stars and junctions (right).

## 5.2 Complex shaped model V2 neurons

Our second experiment represents a comparison to a subset of the results described in Hegde and van Essen [23]. We generated a stimulus set comprising some [23]'s complex shaped stimuli: angles, single bars, tri-stars (three line segments that meet at a point), and arcs/circles, and measured the response of the second layer of our sparse RBM model to these stimuli.[11] We observe that many V2 bases are activated mainly by one of these different stimulus classes. For example, some model V2 neurons activate maximally to single bars; some maximally activate to (acute or obtuse) angles; and others to tri-stars (see Figure 7). Further, the number of V2 bases that are maximally activated by acute angles is significantly larger than the number of obtuse angles, and the number of V2 bases that respond maximally to the tri-stars was much smaller than both preceding cases. This is also consistent with the results described in [23].

## 6 Conclusions

We presented a sparse variant of the deep belief network model. When trained on natural images, this model learns local, oriented, edge filters in the first layer. More interestingly, the second layer captures a variety of both colinear ("contour") features as well as corners and junctions, that in a quantitative comparison to measurements of V2 taken by Ito & Komatsu, appeared to give responses that were similar along several dimensions. This by no means indicates that the cortex is a sparse RBM, but perhaps is more suggestive of contours, corners and junctions being fundamental to the statistics of natural images.[12] Nonetheless, we believe that these results also suggest that sparse

deep learning algorithms, such as our sparse variant of deep belief nets, hold promise for modeling higher-order features such as might be computed in the ventral visual pathway in the cortex.

**Acknowledgments**

We give warm thanks to Minami Ito, Geoffrey Hinton, Chris Williams, Rajat Raina, Narut Sereewattanawoot, and Austin Shoemaker for helpful discussions. Support from the Office of Naval Research under MURI N000140710747 is gratefully acknowledged.

## Footnotes

[1]Due to space constraints, we present an energy function only for the case of real-valued visible units. It is also straightforward to formulate a sparse RBM with binary-valued visible units; for example, we can write the energy function as $E(\mathbf{v}, \mathbf{h}) = -1/\sigma^2(\sum_i c_i v_i + \sum_j b_j h_j + \sum_{i,j} v_i w_{ij} h_j)$ (see also [24]).

[2]Less formally, this regularization ensures that the "firing rate" of the model neurons (corresponding to the latent random variables $h_j$) are kept at a certain (fairly low) level, so that the activations of the model neurons are sparse. Similar intuition was also used in other models (e.g., see Olshausen and Field [10]).

[3]To increase computational efficiency, we made one additional change. Note that the regularization term is defined using a sum over the entire training set; if we use stochastic gradient descent or mini-batches (small subsets of the training data) to estimate this term, it results in biased estimates of the gradient. To ameliorate this, we used mini-batches, but in the gradient step that tries to minimize the regularization term, we update only the bias terms $b_j$'s (which directly control the degree to which the hidden units are activated, and thus their sparsity), instead of updating all the parameters $b_j$ and $w_{ij}$'s.

[4]Downloaded from `http://yann.lecun.com/exdb/mnist/`. Each pixel was normalized to the unit interval, and we used PCA whitening to reduce the dimension to 69 principal components for computational efficiency. (Similar results were obtained without whitening.)

[5]The images were obtained from `http://hlab.phys.rug.nl/imlib/index.html`. We used 100,000 14-by-14 image patches randomly sampled from an ensemble of 2000 images; each subset of 200 patches was used as a mini-batch.

[6]Most other authors' experiments to date using regular (non-sparse) RBMs, when trained on such data, seem to have learned relatively diffuse, unlocalized bases (ones that do not represent oriented edge filters). While sensitive to the parameter settings and requiring a long training time, we found that it is possible in some cases to get a regular RBM to learn oriented edge filter bases as well. But in our experiments, even in these cases we found that repeating this process to build a two layer deep belief net (see Section 4.3) did not encode a significant number of corners/angles, unlike one trained using the sparse RBM; therefore, it showed significantly worse match to the Ito & Komatsu statistics. For example, the fraction of model V2 neurons that respond strongly to a pair of edges near right angles (formally, have peak angle in the range 60-120 degrees) was 2% for the regular RBM, whereas it was 17% for the sparse RBM (and Ito & Komatsu reported 22%). See Section 5.1 for more details.

[7]For the results reported in this paper, we trained the second layer sparse RBM with real-valued visible units; however, the results were very similar when we trained the second layer sparse RBM with binary-valued visible units (except that the second layer weights became less sparse).

[8]The results we report below were very insensitive to the choices of $\sigma$ and $\lambda$. We set $\sigma$ to 0.4 and 0.05 for the first and second layers (chosen to be on the same scale as the standard deviation of the data and the first-layer activations), and $\lambda = 1/p$ in each layer. We used $p = 0.02$ and 0.05 for the first and second layers.

[9]Other details: The stimulus set is created by generating a binary-mask image, that is then scaled to normalize contrast. To determine this scaling constant, we used single bar images by translating and rotating to all possible positions, and fixed the constant such that the top 0.5% (over all translations and rotations) of the stimuli activate the model V1 cells above 0.5. This normalization step corrects for the RBM having been trained on a data distribution (natural images) that had very different contrast ranges than our test stimulus set.

[10]In detail, we generated a set of random low-frequency stimulus, by generating small random KxK (K=2,3,4) images with each pixel drawn from a standard normal distribution, and rescaled the image using bicubic interpolation to 14x14 patches. These stimulus are scaled such that about 5% of the V2 bases fires maximally to these random stimuli. We then exclude the V2 bases that are maximally activated to these random stimuli from the subsequent analysis.

[11]All the stimuli were 14-by-14 pixel image patches. We applied the protocol described in Section 5.1 to the stimulus data, to compute the model V1 and V2 responses.

[12]In preliminary experiments, we also found that when these ideas are applied to self-taught learning [26] (in which one may use unlabeled data to identify features that are then useful for some supervised learning task), using a two-layer sparse RBM usually results in significantly better features for object recognition than using only a one-layer network.

# References

[1] G. E. Hinton, S. Osindero, and Y.-W. Teh. A fast learning algorithm for deep belief nets. *Neural Computation*, 18(7):1527–1554, 2006.

[2] M. Ranzato, C. Poultney, S. Chopra, and Y. LeCun. Efficient learning of sparse representations with an energy-based model. In *NIPS*, 2006.

[3] Y. Bengio, P. Lamblin, D. Popovici, and H. Larochelle. Greedy layer-wise training of deep networks. In *NIPS*, 2006.

[4] H. Larochelle, D. Erhan, A. Courville, J. Bergstra, and Y. Bengio. An empirical evaluation of deep architectures on problems with many factors of variation. In *ICML*, 2007.

[5] G. E. Hinton, S. Osindero, and K. Bao. Learning causally linked MRFs. In *AISTATS*, 2005.

[6] S. Osindero, M. Welling, and G. E. Hinton. Topographic product models applied to natural scene statistics. *Neural Computation*, 18:381–344, 2006.

[7] M. Ito and H. Komatsu. Representation of angles embedded within contour stimuli in area v2 of macaque monkeys. *The Journal of Neuroscience*, 24(13):3313–3324, 2004.

[8] J. H. van Hateren and A. van der Schaaf. Independent component filters of natural images compared with simple cells in primary visual cortex. *Proc.R.Soc.Lond. B*, 265:359–366, 1998.

[9] A. J. Bell and T. J. Sejnowski. The 'independent components' of natural scenes are edge filters. *Vision Research*, 37(23):3327–3338, 1997.

[10] B. A. Olshausen and D. J. Field. Emergence of simple-cell receptive field properties by learning a sparse code for natural images. *Nature*, 381:607–609, 1996.

[11] H. Lee, , A. Battle, R. Raina, and A. Y. Ng. Efficient sparse coding algorithms. In *NIPS*, 2007.

[12] D. Hubel and T. Wiesel. Receptive fields and functional architecture of monkey striate cortex. *Journal of Physiology*, 195:215–243, 1968.

[13] R. L. DeValois, E. W. Yund, and N. Hepler. The orientation and direction selectivity of cells in macaque visual cortex. *Vision Res.*, 22:531–544, 1982a.

[14] H. B. Barlow. The coding of sensory messages. *Current Problems in Animal Behavior*, 1961.

[15] P. O. Hoyer and A. Hyvarinen. A multi-layer sparse coding network learns contour coding from natural images. *Vision Research*, 42(12):1593–1605, 2002.

[16] Y. Karklin and M. S. Lewicki. A hierarchical bayesian model for learning non-linear statistical regularities in non-stationary natural signals. *Neural Computation*, 17(2):397–423, 2005.

[17] A. Hyvarinen and P. O. Hoyer. Emergence of phase and shift invariant features by decomposition of natural images into independent feature subspaces. *Neural Computation*, 12(7):1705–1720, 2000.

[18] A. Hyvärinen, P. O. Hoyer, and M. O. Inki. Topographic independent component analysis. *Neural Computation*, 13(7):1527–1558, 2001.

[19] A. Hyvarinen, M. Gutmann, and P. O. Hoyer. Statistical model of natural stimuli predicts edge-like pooling of spatial frequency channels in v2. *BMC Neuroscience*, 6:12, 2005.

[20] L. Wiskott and T. Sejnowski. Slow feature analysis: Unsupervised learning of invariances. *Neural Computation*, 14(4):715–770, 2002.

[21] G. Boynton and J. Hegde. Visual cortex: The continuing puzzle of area v2. *Current Biology*, 14(13):R523–R524, 2004.

[22] J. B. Levitt, D. C. Kiper, and J. A. Movshon. Receptive fields and functional architecture of macaque v2. *Journal of Neurophysiology*, 71(6):2517–2542, 1994.

[23] J. Hegde and D.C. Van Essen. Selectivity for complex shapes in primate visual area v2. *Journal of Neuroscience*, 20:RC61–66, 2000.

[24] G. E. Hinton and R. R. Salakhutdinov. Reducing the dimensionality of data with neural networks. *Science*, 313(5786):504–507, 2006.

[25] G. E. Hinton. Training products of experts by minimizing contrastive divergence. *Neural Computation*, 14:1771–1800, 2002.

[26] R. Raina, A. Battle, H. Lee, B. Packer, and A. Y. Ng. Self-taught learning: Transfer learning from unlabeled data. In *ICML*, 2007.

